# Semi-supervised Learning in Gigantic Image Collections

**Rob Fergus**
Courant Institute, NYU,
715 Broadway,
New York, NY 10003
fergus@cs.nyu.edu

**Yair Weiss**
School of Computer Science,
Hebrew University,
91904, Jerusalem, Israel
yweiss@huji.ac.il

**Antonio Torralba**
CSAIL, EECS, MIT,
32 Vassar St.,
Cambridge, MA 02139
torralba@csail.mit.edu

## Abstract

With the advent of the Internet it is now possible to collect hundreds of millions of images. These images come with varying degrees of label information. "Clean labels" can be manually obtained on a small fraction, "noisy labels" may be extracted automatically from surrounding text, while for most images there are no labels at all. Semi-supervised learning is a principled framework for combining these different label sources. However, it scales polynomially with the number of images, making it impractical for use on gigantic collections with hundreds of millions of images and thousands of classes. In this paper we show how to utilize recent results in machine learning to obtain highly efficient approximations for semi-supervised learning that are *linear* in the number of images. Specifically, we use the convergence of the eigenvectors of the normalized graph Laplacian to eigenfunctions of weighted Laplace-Beltrami operators. Our algorithm enables us to apply semi-supervised learning to a database of 80 million images gathered from the Internet.

## 1   Introduction

Gigantic quantities of visual imagery are present on the web and in off-line databases. Effective techniques for searching and labeling this ocean of images and video must address two conflicting problems: (i) the techniques to understand the visual content of an image and (ii) the ability to scale to millions of billions of images or video frames. Both aspects have received significant attention from researchers, the former being addressed by recent work on object and scene recognition, while the latter is the focus of the content-based image retrieval community (CBIR) [7]. A key issue pertaining to both aspects of the problem is the diversity of label information accompanying real world image data. A variety of collaborative and online annotation efforts have attempted to build large collections of human labeled images, ranging from simple image classifications, to bounding-boxes and precise pixel-level segmentation [16, 21, 24]. While impressive, these manual efforts have no hope of scaling to the many billions of images on the Internet. However, even though most images on the web lack human annotation, they often have some kind of noisy label gleaned from nearby text or from the image filename and often this gives a strong cue about the content of the image. Finally, there are images where we have no information beyond the pixels themselves. Semi-supervised learning (SSL) methods are designed to handle this spectrum of label information [26, 28]. They rely on the density structure of the data itself to propagate known labels to areas lacking annotations, and provide a natural way to incorporate labeling uncertainty. However, to model the density of the data, each point must measure its proximity to every other. This requires polynomial time – prohibitive for large-scale problems.

In this paper, we introduce a semi-supervised learning scheme that is linear in the number of images, enabling us to tackle very large scale problems. Building on recent results in spectral graph theory, we efficiently construct accurate numerical approximations to the eigenvectors of the normalized graph Laplacian. Using these approximations, we can easily propagate labels through huge collections of images.

## 1.1 Related Work

Cleaning up Internet image data has been explored by several authors: Berg *et al.* [4], Fergus *et al.* [8], Li *et al.* [13], Vijayanarasimhan *et al.* [22], amongst others. Unlike our approach, these methods operate independently on each class and would be problematic to scale to millions or billions of images. A related group of techniques use active labeling, e.g. [10]. Semi-supervised learning is a rapidly growing sub-field of machine learning, dealing with datasets that have a large number of unlabeled points and a much smaller number of labeled points (see [5] for a recent overview). The most popular approaches are based on the graph Laplacian (e.g. [26, 28] and there has been much theoretical work devoted to the asymptotics of these Laplacians [3, 6, 14]. However, these methods require the explicit manipulation of an $n \times n$ Laplacian matrix ($n$ being the number of data points), for example [2] notes: "our algorithms compute the inverse of a dense Gram matrix which leads to $O(n^3)$ complexity. This may be impractical for large datasets."

The large computational complexity of standard graph Laplacian methods has lead to a number of recent papers on efficient semi-supervised learning (see [27] for an overview). Many of these methods (e.g. [18, 12, 29, 25] are based on calculating the Laplacian only for a smaller, backbone, graph which reduces complexity to be cubic in the size of the small graph. In most cases [18, 12] the smaller graph is built simply by randomly subsampling a subset of the points, while in [29] a mixture model is learned on the original dataset and each mixture component defines a node in the backbone graph. In [25] the backbone graph is found using non-negative matrix factorization. In [9] the backbone graph is a uniform grid over the high dimensional space (so the number of nodes grows exponentially with dimension). In [20] the number of datapoints is not reduced but rather the number of edges. This allows the use of sparse numerical algebra techniques.

The problem with approaches based on backbone graphs is that the spectrum of the graph Laplacian can change dramatically with different backbone construction methods [12]. This can also be seen visually (see Fig. 3) by examining the clusterings suggested by the full data and a small subsample. Even in cases where the "correct" clustering is obvious when the full data is considered, the smaller subset may suggest erroneous clusterings (e.g. Fig. 3(left)). In our approach, we take an alternative route. Rather than trying to reduce the number of points, we take the limit as the number of points goes to infinity.

## 2 Semi-supervised Learning

We start by introducing semi-supervised learning in a graph setting and then describe an approximation that reduces the learning time from polynomial to linear in the number of images. Fig. 1 illustrates the semi supervised learning problem. Following the notations of Zhu *et al.* [28], we are given a labeled dataset of input-output pairs $(X_l, Y_l) = \{(x_1, y_1), ..., (x_l, y_l)\}$ and an unlabeled dataset $X_u = \{x_{l+1}, ..., x_n\}$. Thus in Fig. 1(a) we are given two labeled points and 500 unlabeled points. Fig. 1(b) shows the output of a nearest neighbor classifier on the unlabeled points. The purely supervised solution ignores the apparent clustering suggested by the data.

In order to use the unlabeled data, we form a graph $G = (V, E)$ where the vertices $V$ are the datapoints $x_1, ..., x_n$, and the edges $E$ are represented by an $n \times n$ matrix $W$. Entry $W_{ij}$ is the edge weight between nodes $i, j$ and a common practice is to set $W_{ij} = \exp(-\|x_i - x_j\|^2/2\epsilon^2)$. Let $D$ be a diagonal matrix whose diagonal elements are given by $D_{ii} = \sum_j W_{ij}$, the combinatorial graph Laplacian is defined as $L = D - W$, which is also called the unnormalized Laplacian.

In graph-based semi-supervised learning, the graph Laplacian $L$ is used to define a smoothness operator that takes into account the unlabeled data. The main idea is to find functions $f$ which agree with the labeled data but are also *smooth* with respect to the graph. The smoothness is measured by the graph Laplacian:

$$f^T L f = \frac{1}{2} \sum_{i,j} W_{ij} \left( f(i) - f(j) \right)^2$$

Of course simply minimizing smoothness can be achieved by the trivial solution $f = 1$, but in semi-supervised learning, we minimize a combination of the smoothness and the training loss. For squared error training loss, this is simply:

$$J(f) = f^T L f + \sum_{i=1}^{l} \lambda(f(i) - y_i)^2 = f^T L f + (f - y)^T \Lambda (f - y)$$

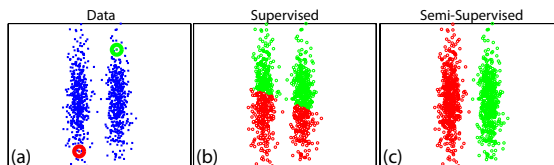

Figure 1: Comparison of supervised and semi-supervised learning on toy data. Semi-supervised learning seeks functions that are smooth with respect to the input density.

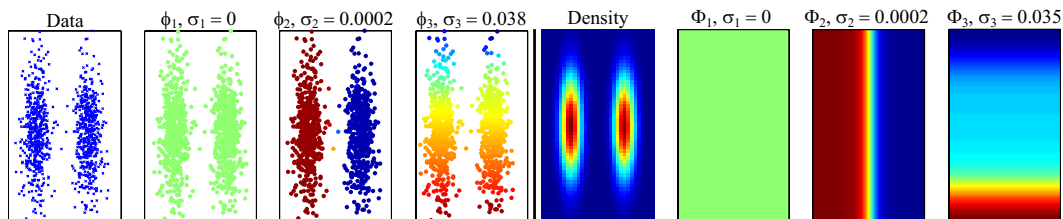

Figure 2: **Left:** The three generalized eigenvectors of the graph Laplacian, for the toy data. Note that the semi-supervised solution can be written as a linear combination of these eigenvectors (in this case, the second eigenvector is enough). Using generalized eigenvectors (or equivalently normalized Laplacians) increases robustness of the first eigenvectors, compared to using the un-normalized eigenvectors. **Right:** The 2D density of the toy data, and the associated smoothness eigenfunctions defined by that density. The plots use the Matlab jet colormap.

where $\Lambda$ is a diagonal matrix whose diagonal elements are $\Lambda_{ii} = \lambda$ if $i$ is a labeled point and $\Lambda_{ii} = 0$ for unlabeled points. The minimizer is of course a solution to $(L + \Lambda)f = \Lambda y$. Fig. 1(c) shows the semi-supervised solution.

Although the solution can be given in closed form for the squared error loss, note that it requires solving an $n \times n$ system of linear equations. For large $n$ this poses serious problems with computation time and robustness. But as suggested in [5, 17, 28], the dimension of the problem can be reduced dramatically by only working with a small number of eigenvectors of the Laplacian.

Let $\Phi_i, \sigma_i$ be the generalized eigenvectors and eigenvalues of the graph Laplacian $L$ (solutions to $L\phi_i = \sigma_i D\phi_i$). Note that the smoothness of an eigenvector $\Phi_i$ is simply $\Phi_i^T L \Phi_i = \sigma_i$ so that eigenvectors with smaller eigenvalues are smoother. Since any vector in $R^n$ can be written $f = \sum_i \alpha_i \Phi_i$, the smoothness of a vector is simply $\sum_i \alpha_i^2 \sigma_i$ so that smooth vectors will be linear combinations of the eigenvectors with small eigenvalues[1].

Fig. 2(left) shows the three generalized eigenvectors of the Laplacian for the toy data shown in Fig. 1(a). Note that the semi-supervised solution (Fig. 1(c)) is a linear combination of these three eigenvectors (in fact just one eigenvector is enough). In general, we can significantly reduce the dimension of $f$ by requiring it to be of the form $f = U\alpha$ where $U$ is a $n \times k$ matrix whose columns are the $k$ eigenvectors with smallest eigenvalue. We now have:

$$J(\alpha) = \alpha^T \Sigma \alpha + (U\alpha - y)^T \Lambda (U\alpha - y)$$

The minimizing $\alpha$ is now a solution to the $k \times k$ system of equations:

$$(\Sigma + U^T \Lambda U)\alpha = U^T \Lambda y \tag{1}$$

## 2.1 From Eigenvectors to Eigenfunctions

Given the eigenvectors of the graph Laplacian, we can now solve the semi-supervised problem in a reduced dimensional space. But to find the eigenvectors in the first place, we need to diagonalize a $n \times n$ matrix. How can we efficiently calculate the eigenvectors as the number of unlabeled points increases?

We follow [23, 14] in assuming the data $x_i \in R^d$ are samples from a distribution $p(x)$ and analyzing the eigenfunctions of the smoothness operator defined by $p(x)$. Fig. 2(right) shows the density in two

dimensions for the toy data. This density defines a weighted smoothness operator on any function $F(x)$ defined on $R^d$ which we will denote by $L_p(F)$:

$$L_p(F) = \frac{1}{2} \int (F(x_1) - F(x_2))^2 W(x_1, x_2) p(x_1) p(x_2) dx_1 x_2$$

with $W(x_1, x_2) = \exp(-\|x_1 - x_2\|^2 / 2\epsilon^2)$. Just as the graph Laplacian defined eigenvectors of increasing smoothness, the smoothness operator will define eigenfunctions of increasing smoothness. We define the first eigenfunction of $L_P(f)$ by a minimization problem:

$$\Phi_1 = \arg \min_{F: \int F^2(x) p(x) D(x) dx = 1} L_p(F)$$

where $D(x) = \int_{x_2} W(x, x_2) p(x_2) dx_2$. Note that the first eigenfunction will always be the trivial function $\Phi(x) = 1$ since it has maximal smoothness $L_P(1) = 0$. The second eigenfunction of $L_p(f)$ minimizes the same problem, with the additional constraint that it be orthogonal to the first eigenfunction $\int F(x)\Phi_1(x) D(x) p(x) dx = 0$. More generally, the $k$th eigenfunction minimizes $L_p(f)$ under additional constraints that $\int F(x)\Phi_l(x) p(x) D(x) dx = 0$ for all $l < k$. The eigenvalue of an eigenfunction $\Phi_k$ is simply its smoothness $\sigma_k = L_p(\Phi_k)$. Fig. 2(right) shows the first three eigenfunctions corresponding to the density of the toy data. Similar to the eigenvectors of the graph Laplacian, the second eigenfunction reveals the natural clustering of the data. Note that the eigenvalue of the eigenfunctions is similar to the eigenvalue of the discrete generalized eigenvector.

How are these eigenfunctions related to the generalized eigenvectors of the Laplacian? It is easy to see that as $n \to \infty$, $\frac{1}{n^2} f^T L f = \frac{1}{2n^2} \sum_{i,j} W_{ij} (f(i) - f(j))^2$ will approach $L_p(F)$, and $\frac{1}{n} \sum_i f^2(i) D(i, i)$ will approach $\int F^2(x) D(x) p(x) dx$ so that the minimization problems that define the eigenvectors approach the problems that define the eigenfunctions as $n \to \infty$. Thus under suitable convergence conditions, the eigenfunctions can be seen as the limit of the eigenvectors as the number of points goes to infinity [1, 3, 6, 14]. For certain parametric probability functions (e.g. uniform, Gaussian) the eigenfunctions can be calculated analytically [14, 23]. Thus for these cases, there is a tremendous advantage in estimating $p(x)$ and calculating the eigenfunctions from $p(x)$ rather than attempting to estimate the eigenvectors directly. For example, consider a problem with 80 million datapoints sampled from a 32 dimensional Gaussian. Instead of diagonalizing an 80 million by 80 million matrix, we can simply estimate a $32 \times 32$ covariance matrix and get analytical eigenfunctions. In low dimensions, we can calculate the eigenfunction numerically by discretizing the density. Let $g$ be the eigenfunction values at a set of discrete points, then $g$ satisfies:

$$(\tilde{D} - P\tilde{W}P)g = \sigma P\hat{D}g \tag{2}$$

where $\tilde{W}$ is the affinity between the discrete points, $P$ is a diagonal matrix whose diagonal elements give the density at the discrete points, and $\tilde{D}$ is a diagonal matrix whose diagonal elements are the sum of the columns of $P\tilde{W}P$, and $\hat{D}$ is a diagonal matrix whose diagonal elements are the sum of the columns of $P\tilde{W}$. This method was used to calculate the eigenfunctions in Fig. 2(right).

Instead of assuming that $p(x)$ has a simple, parametric form, we will use a more modest assumption, that $p(x)$ has a product form. Specifically, we assume that if we rotate the data $s = Rx$ then $p(s) = p(s_1)p(s_2)\cdots p(s_d)$. This assumption allows us to calculate the eigenfunctions of $L_p$ using only the marginal distributions $p(s_i)$.

**Observation:** Assume $p(s) = p(s_1)p(s_2)\cdots p(s_d)$. Let $p_k$ be the marginal distribution of a single coordinate in $s$. Let $\Phi_i(s_k)$ be an eigenfunction of $L_{p_k}$ with eigenvalue $\sigma_i$, then $\Phi_i(s) = \Phi_i(s_k)$ is also an eigenfunction of $L_p$ with the same eigenvalue $\sigma_i$.

**Proof:** This follows from the observation in [14, 23] that for separable distributions, the eigenfunctions are also separable.

This observation motivates the following algorithm:

- Find a rotation of the data $R$, so that $s = Rx$ are as independent as possible.
- For each "independent" component $s_k$, use a histogram to approximate the density $p(s_k)$. In order to regularize the solution (see below), we add a small constant to the value of the histogram at each bin.

- Given the approximated density $p(s_k)$, solve numerically for eigenfunctions and eigenvalues of $L_{p_k}$ using Eqn. 2. As discussed above, this can be done by solving a generalized eigenvalue problem for a $B \times B$ matrix, where $B$ is the number of bins in the histogram.
- Order the eigenfunctions from all components by increasing eigenvalue.

The need to add a small constant to the histogram comes from the fact that the smoothness operator $L_p(F)$ ignores the value of $F$ wherever the density vanishes, $p(x) = 0$. Thus the eigenfunctions can oscillate wildly in regions with zero density. By adding a small constant to the density we enforce an additional smoothness regularizer, even in regions of zero density. Similar regularizers are used in [2, 9].

This algorithm will recover eigenfunctions of $L_p$, which depend only on a single coordinate. As discussed in [23], products of these eigenfunctions for different coordinates are also eigenfunctions, but we will assume the semi-supervised solution is a linear combination of only the single-coordinate eigenfunctions. By choosing the $k$ eigenfunctions with smallest eigenvalue we now have $k$ functions $\Phi_k(x)$ whose value is given at a set of discrete points for each coordinate. We then use linear interpolation in 1D to interpolate $\Phi(x)$ at each of the labeled points $x_l$. This allows us to solve Eqn. 1 in time that is *independent of the number of unlabeled points*.

Although this algorithm has a number of approximate steps, it should be noted that if the "independent" components are indeed independent, and if the semi-supervised solution is only a linear combination of the single-coordinate eigenfunctions, then this algorithm will exactly recover the semi-supervised solution as $n \to \infty$. Consider again a dataset of 80 million points in 32 dimensions and assume 100 bins per dimension. If the independent components $s = Rx$ are indeed independent, then this algorithm will exactly recover the semi-supervised solution by solving 32 $100 \times 100$ generalized eigenvector problems and a single $k \times k$ least squares problem. In contrast, directly estimating the eigenvectors of the graph Laplacian will require diagonalizing an 80 million by 80 million matrix.

## 3 Experiments

In this section we describe experiments to illustrate the performance and scalability of our approach. The results will be reported on the Tiny Images database [19], in combination with the CIFAR-10 label set [11]. This data is diverse and highly variable, having been collected directly from Internet search engines. The set of labels allows us to accurately measure the performance of our algorithm, while using data typical of the large-scale Internet settings for which our algorithm is designed.

We start with a toy example that illustrates our eigenfunction approach, compared to the Nystrom method of Talwalker *et al.* [18], another approximate semi-supervised learning scheme that can scale to large datasets. In Fig. 3 we show two different 2D datasets, designed to reveal the failure modes of the two methods.

### 3.1 Features

For the experiments in this paper we use global image descriptors to represent the entire image (there is no attempt to localize the objects within the images). Each image is thus represented by a single Gist descriptor [15], which we then project down to 64 dimensions using PCA. As

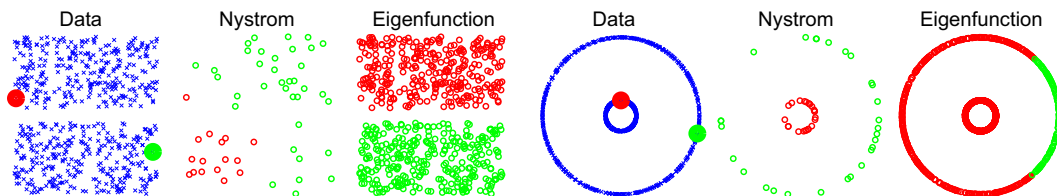

Figure 3: A comparison of the separable eigenfunction approach and the Nystrom method. Both methods have comparable computational cost. The Nystrom method is based on computing the graph Laplacian on a set of sparse landmark points and fails in cases where the landmarks do not adequately summarize the density (left). The separable eigenfunction approach fails when the density is far from a product form (right).

illustrated in Fig. 3, the eigenfunction approach assumes that the input distribution is separable over dimension. In Fig. 4 we show that while the raw gist descriptors exhibit strong dependencies between dimensions, this is no longer the case after the PCA projection. Note that PCA is one of the few types of projection permitted: since distances between points must be preserved only rotations of the data are allowed.

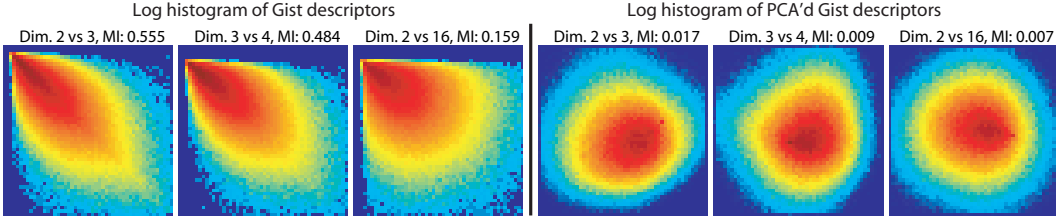

Figure 4: 2D log histograms formed from 1 million Gist descriptors. Red and blue correspond to high and low densities respectively. **Left:** three pairs of dimensions in the raw Gist descriptor, along with their mutual information score (MI), showing strong dependencies between dimensions. **Right:** the dimensions in the Gist descriptors after a PCA projection, as used in our experiments. The dependencies between dimensions are now much weaker, as the MI scores show. Hence the separability assumption made by our approach is not an unreasonable one for this type of data.

### 3.2  Experiments with CIFAR label set

The CIFAR dataset [11] was constructed by asking human subjects to label a subset of classes of the Tiny Images dataset. For a given keyword and image, the subjects determined whether the given image was indeed an image of that keyword. The resulting labels span 386 distinct keywords in the Tiny Images dataset. Our experiments use the sub-set of 126 classes which had at least 200 positive labels and 300 negative labels, giving a total of 63,000 images.

Our experimental protocol is as follows: we take a random subset of $C$ classes from the set of 126. For each class $c$, we randomly choose a fixed test-set of 100 positive and 200 negative examples, reflecting the typical signal-to-noise ratio found in images from Internet search engines. The training examples consist of $t$ positive/negative pairs drawn from the remaining pool of 100 positive/negative images for each keyword.

For each class in turn, we use our scheme to propagate labels from the training examples to the test examples. By assigning higher probability (values in $f$) to the genuine positive images of each class, we are able to re-rank the images. We also make use of the the training examples from keywords other than $c$ by treating them as additional negative examples. For example, if we have $C = 16$ keywords and $t = 5$ training pairs per keyword, then we have 5 positive training examples and (5+(16-1)*10)=155 negative training examples for each class. We use these to re-rank the 300 test images of that particular class. Note that the propagation from labeled images to test images may go through the unlabeled images that are not even in the same class. Our use of examples from other classes as negative examples is motivated by real problems, where training labels are spread over many keywords but relatively few labels are available per class.

In experiments using our eigenfunction approach, we compute a fixed set of $k$=256 eigenfunctions on the entire 63,000 datapoints in the 64D space with $\epsilon = 0.2$ and used these for all runs. For approaches that require explicit formation of the affinity matrix, we calculate the distance between the 64D image descriptors using $\epsilon = 0.125$. All approaches use $\lambda = 50$. To evaluate performance, we choose to measure the precision at a low recall rate of 15%, this being a sensible operating point as it corresponds to the first webpage or so in an Internet retrieval setting. Given the split of +ve/-ve examples in the test data, chance level performance corresponds to a precision of 33%. All results were generated by averaging over 10 different runs, each with different random train/test draws, and with different subsets of classes.

In our first set of experiments, shown in Fig. 5(left), we compare our eigenfunction approach to a variety of alternative learning schemes. We use $C = 16$ different classes drawn randomly from the 126, and vary the number of training pairs $t$ from 0 up to 100 (thus the total number of labeled points, positive and negative, varied from 0 to 3200). Our eigenfunction approach outperforms other methods, particularly where relatively few training examples are available. We use two baseline classifiers: (i) Nearest-Neighbor and (ii) RBF kernel SVM, with kernel width $\epsilon$. The SVM approach

badly over-fits the data for small numbers of training examples, but catches up with the eigenfunction approach once 64+ve/1984-ve labeled examples are used.

We also test a range of SSL approaches. The exact least-squares approach ($f = (L + \Lambda)^{-1}\Lambda Y$) achieves comparable results to the eigenfunction method, although it is far more expensive. The eigenvector approach (Eqn. 1) performs less well, being limited by the $k = 256$ eigenvectors used (as $k$ is increased, the performance converges to the exact least-squares solution). Neither of these methods scale to large image collections as the affinity matrix $W$ becomes too big and cannot be inverted or have its eigenvectors computed. Fig. 5(left) also shows the efficient Nystrom method [18], using 1000 landmark points, which has a somewhat disappointing performance. Evidently, as in Fig. 3, the landmark points do not adequately summarize the density. As the number of landmarks is increased, the performance approaches that of the least squares solution.

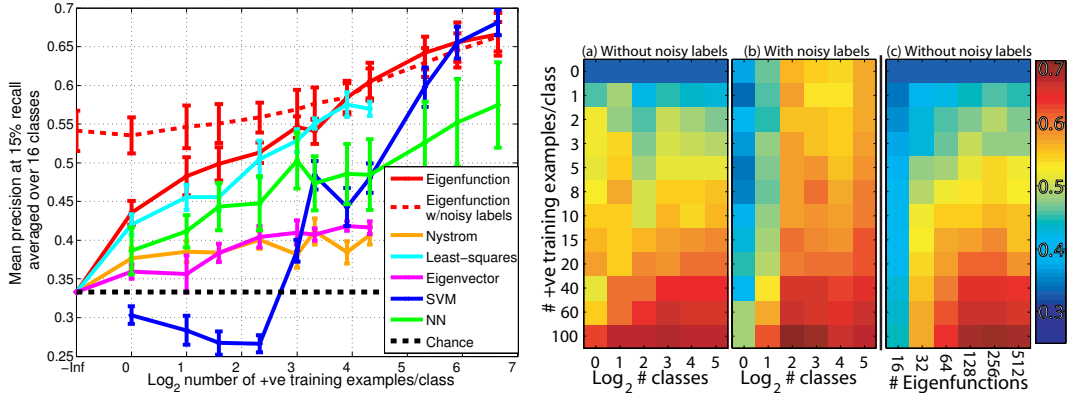

Figure 5: **Left:** Performance (precision at 15% recall) on the Tiny Image CIFAR label set for different learning schemes as the number of training pairs is increased, averaged over 16 different classes. -Inf corresponds to the unsupervised case (0 examples). Our eigenfunction scheme (solid red) outperforms standard supervised methods (nearest-neighbors (green) and a Gaussian SVM (blue)) for small numbers of training pairs. Compared to other semi-supervised schemes, ours matches the exact least squares solution (which is too expensive to run on a large number of images), while outperforming approximate schemes, such as Nystrom [18]. By using noisy labels in addition to the training pairs, the performance is boosted when few training examples are available (dashed red). **Right: (a)**: The performance of our eigenfunction approach as the number of training pairs per class and number of classes is varied. Increasing the number of classes also aids performance since labeled examples from other classes can be used as negative examples. **(b)**: As for **(a)** but now using noisy label information (Section 3.3). Note the improvement in performance when few training pairs are available. **(c)**: The performance of our approach (using no noisy labels) as the number of eigenfunctions is varied.

In Fig. 5(right)(a) we explore how our eigenfunction approach performs as the number of classes $C$ is varied, for different numbers of training pairs $t$ per class. For a fixed $t$, as $C$ increases, the number of negative examples available increases thus aiding performance. Fig. 5(right)(c) shows the effect of varying the number of eigenfunctions $k$ for $C = 16$ classes. The performance is fairly stable above $k = 128$ eigenfunctions (i.e. on average 2 per dimension), although some mild over-fitting seems to occur for small numbers of training examples when a very large number is used.

### 3.3 Leveraging noisy labels

In the experiments above, only two types of data are used: labeled training examples and unlabeled test examples. However, an additional source is the noisy labels from the Tiny Image dataset (the keyword used to query the image search engine). These labels can easily be utilized by our framework: all 300 test examples for a class $c$ are given a positive label with a small weight ($\lambda/10$), while the $300(C-1)$ test examples from other classes are given negative label with the same small weight. Note that these labels do not reveal any information about which of the 300 test images are true positives. These noisy labels can provide a significant performance gain when few training (clean) labels are available, as shown in Fig. 5(left) (c.f. solid and dashed red lines). Indeed, when no training labels are available, just the noisy labels, our eigenfunction scheme still performs very well. The performance gain is explored in more detail in Fig. 5(right)(b). In summary, using

the eigenfunction approach with noisy labels, the performance obtained with a total of 32 labeled examples is comparable to the SVM trained with 64*16=512 labeled examples.

### 3.4 Experiments on Tiny Images dataset

Our final experiment applies the eigenfunction approach to the whole of the Tiny Images dataset (79,302,017 images). We map the gist descriptor for each image down to a 32D space using PCA and precompute $k = 64$ eigenfunctions over the entire dataset. The 445,954 CIFAR labels (64,185 of which are +ve) cover 386 keywords, any of which can be re-ranked by solving Eqn. 1, which takes around 1ms on a fast PC. In Fig. 6 we show our scheme on four different keywords, each using 3 labeled training pairs, resulting in a significant improvement in quality over the original ordering. A nearest-neighbor classifier which is not regularized by the data density performs worse than our approach.

Ranking from search engine    Nearest Neighbor re-ranking    Eigenfunction re-ranking

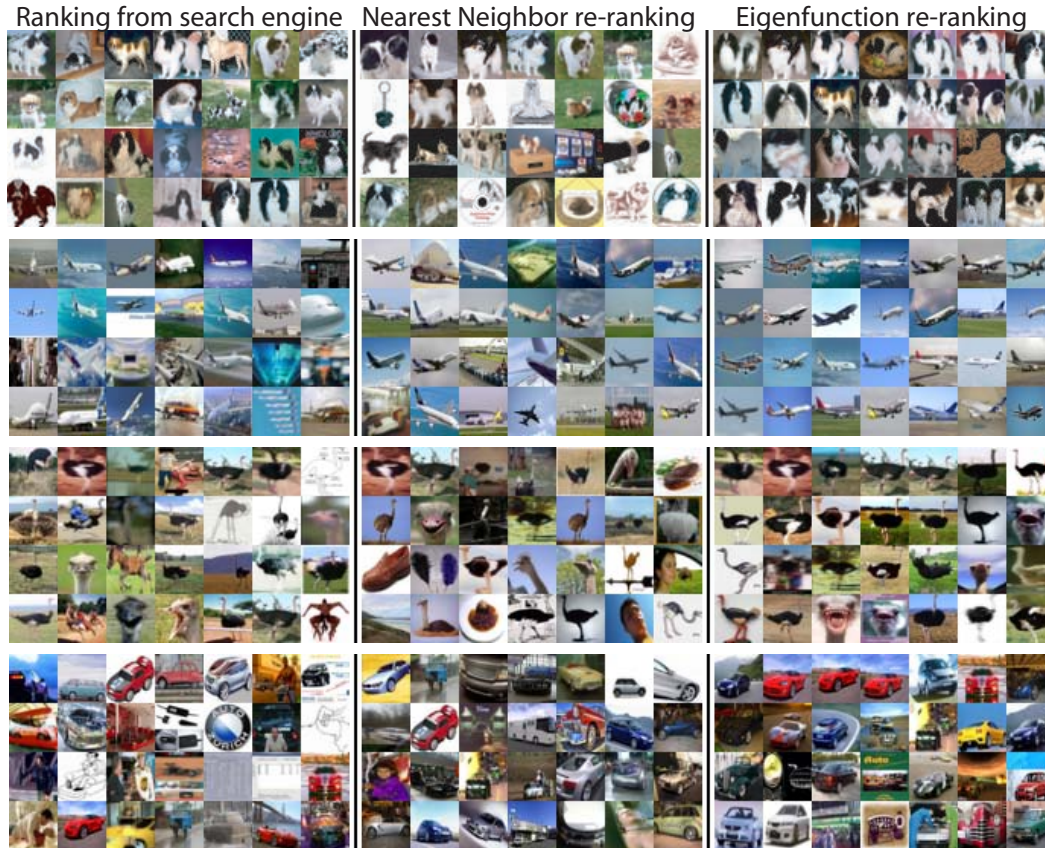

Figure 6: Re-ranking images from 4 keywords in an 80 million image dataset, using 3 labeled pairs for each keyword. Rows from top: "Japanese spaniel", "airbus", "ostrich", "auto". From L to R, the columns show the original image order, results of nearest-neighbors and the results of our eigenfunction approach. By regularizing the solution using eigenfunctions computed from all 80 million images, our semi-supervised scheme outperforms the purely supervised method.

## 4 Discussion

We have proposed a novel semi-supervised learning scheme that is linear in the number of images, and then demonstrated it on challenging datasets, including one of 80 million images. The approach can easily be parallelized making it practical for Internet-scale image collections. It can also incorporate a variety of label types, including noisy labels, in one consistent framework.

### Acknowledgments
The authors would like to thank Héctor Bernal and the anonymous reviewers and area chairs for their constructive comments. We also thank Alex Krizhevsky and Geoff Hinton for providing the CIFAR label set. Funding support came from: NSF Career award (ISI 0747120), ISF and a Microsoft Research gift.

## Footnotes

[1]This discussion holds for both ordinary and generalized eigenvectors, but the latter are much more stable and we use them.

# References

[1] M. Belkin and P. Niyogi. Towards a theoretical foundation for laplacian based manifold methods. *Journal of Computer and System Sciences*, 2007.

[2] M. Belkin, P. Niyogi, and V. Sindhwani. Manifold regularization: A geometric framework for learning from labeled and unlabeled examples. *JMLR*, 7:2399–2434, 2006.

[3] Y. Bengio, O. Delalleau, N. L. Roux, J.-F. Paiement, P. Vincent, and M. Ouimet. Learning eigenfunctions links spectral embedding and kernel PCA. In *NIPS*, pages 2197–2219, 2004.

[4] T. Berg and D. Forsyth. Animals on the web. In *CVPR*, pages 1463–1470, 2006.

[5] O. Chapelle, B. Schölkopf, and A. Zien. *Semi-Supervised Learning*. MIT Press, 2006.

[6] R. R. Coifman, S. Lafon, A. Lee, M. Maggioni, B. Nadler, F. Warner, and S. Zucker. Geometric diffusion as a tool for harmonic analysis and structure definition of data, part i: Diffusion maps. *PNAS*, 21(102):7426–7431, 2005.

[7] R. Datta, D. Joshi, J. Li, and J. Z. Wang. Image retrieval: Ideas, influences, and trends of the new age. *ACM Computing Surveys*, 2008.

[8] R. Fergus, L. Fei-Fei, P. Perona, and A. Zisserman. Learning object categories from google's image search. In *ICCV*, volume 2, pages 1816–1823, Oct. 2005.

[9] J. Garcke and M. Griebel. Semi-supervised learning with sparse grids. In *ICML workshop on learning with partially classified training data*, 2005.

[10] A. Kapoor, K. Grauman, R. Urtasun, and T. Darrell. Active learning with gaussian processes for object categorization. In *CVPR*, 2007.

[11] A. Krizhevsky and G. E. Hinton. Learning multiple layers of features from tiny images. Technical report, Computer Science Department, University of Toronto, 2009.

[12] S. Kumar, M. Mohri, and A. Talwalkar. Sampling techniques for the Nystrom method. In *AISTATS*, 2009.

[13] L. J. Li, G. Wang, and L. Fei-Fei. Optimol: automatic object picture collection via incremental model learning. In *CVPR*, 2007.

[14] B. Nadler, S. Lafon, R. R. Coifman, and I. G. Kevrekidis. Diffusion maps, spectral clustering and reaction coordinates of dynamical systems. *Applied and Computational Harmonic Analysis*, 21:113–127, 2006.

[15] A. Oliva and A. Torralba. Modeling the shape of the scene: a holistic representation of the spatial envelope. *IJCV*, 42:145–175, 2001.

[16] B. C. Russell, A. Torralba, K. P. Murphy, and W. T. Freeman. Labelme: a database and web-based tool for image annotation. *IJCV*, 77(1):157–173, 2008.

[17] B. Schoelkopf and A. Smola. *Learning with Kernels Support Vector Machines, Regularization, Optimization, and Beyond*. MIT Press,, 2002.

[18] A. Talwalkar, S. Kumar, and H. Rowley. Large-scale manifold learning. In *CVPR*, 2008.

[19] A. Torralba, R. Fergus, and W. T. Freeman. 80 million tiny images: a large database for non-parametric object and scene recognition. *IEEE PAMI*, 30(11):1958–1970, November 2008.

[20] I. Tsang and J. Kwok. Large-scale sparsified manifold regularization. In *NIPS*, 2006.

[21] L. van Ahn. The ESP game, 2006.

[22] S. Vijayanarasimhan and K. Grauman. Keywords to visual categories: Multiple-instance learning for weakly supervised object categorization. In *CVPR*, 2008.

[23] Y. Weiss, A. Torralba, and R. Fergus. Spectral hashing. In *NIPS*, 2008.

[24] B. Yao, X. Yang, and S. C. Zhu. Introduction to a large scale general purpose ground truth dataset: methodology, annotation tool, and benchmarks. In *EMMCVPR*, 2007.

[25] K. Yu, S. Yu, and V. Tresp. Blockwise supervised inference on large graphs. In *ICML workshop on learning with partially classified training data*, 2005.

[26] D. Zhou, O. Bousquet, T. N. Lal, J. Weston, and B. Schölkopf. Learning with local and global consistency. In *NIPS*, 2004.

[27] X. Zhu. Semi-supervised learning literature survey. Technical Report 1530, University of Wisconsin Madison, 2008.

[28] X. Zhu, Z. Ghahramani, and J. Lafferty. Semi-supervised learning using gaussian fields and harmonic functions. In *In ICML*, pages 912–919, 2003.

[29] X. Zhu and J. Lafferty. Harmonic mixtures: combining mixture models and graph-based methods for inductive and scalable semi-supervised learning. In *ICML*, 2005.

